# Estimating Internal Variables and Parameters of a Learning Agent by a Particle Filter

**Kazuyuki Samejima**  **Kenji Doya**
Department of Computational Neurobiology
ATR Computational Neuroscience laboratories;
"Creating the Brain", CREST, JST.
"Keihan-na Science City", Kyoto, 619-0288, Japan
{samejima, doya}@atr.jp

**Yasumasa Ueda**  **Minoru Kimura**
Department of Physiology, Kyoto Prefecture University of Medicine,
Kyoto, 602-8566, Japan
{yasu, mkimura}@basic.kpu-m.ac.jp

## Abstract

When we model a higher order functions, such as learning and memory, we face a difficulty of comparing neural activities with hidden variables that depend on the history of sensory and motor signals and the dynamics of the network. Here, we propose novel method for estimating hidden variables of a learning agent, such as connection weights from sequences of observable variables. Bayesian estimation is a method to estimate the posterior probability of hidden variables from observable data sequence using a dynamic model of hidden and observable variables. In this paper, we apply particle filter for estimating internal parameters and meta-parameters of a reinforcement learning model. We verified the effectiveness of the method using both artificial data and real animal behavioral data.

## 1   Introduction

In neurophysiology, the traditional approach to discover unknown information processing mechanisms is to compare neuronal activities with external variables, such as sensory stimuli or motor output. Recent advances in computational neuroscience allow us to make predictions on neural mechanisms based on computational models. However, when we model higher order functions, such as attention, memory and learning, the model must inevitably include hidden variables which are difficult to infer directly from externally observable variables.

Although the assessment of the plausibility of such models depends on the right estimate of the hidden variables, tracking their values in an experimental setting is a difficult problem. For example, in learning agents, hidden variables such as connection weights change in time. In addition, the course of learning is modulated by hidden meta-parameters such as

the learning rate.

The goal of this study is two-fold: First to establish a method to estimate hidden variables, including meta-parameters from observable experimental data. Second to provide a method for objectively selecting the most plausible computational model out of multiple candidates. We introduce a numerical Bayesian estimation method, known as particle filtering, to estimate hidden variables. We validate this method with a reinforcement learning task.

## 2 Reinforcement learning model as an animal and a human decision processes

Reinforcement learning can be a model of animal or human behaviors based on reward delivery. Notably, the response of monkey midbrain dopamine neurons are successfully explained by the temporal differnce (TD) error of reinforcement learning models [2]. The goal of reinforcement learning is to improve the policy so that the agent maximizes rewards in the long run. The basic strategy of reinforcement learning is to estimate cumulative future reward under the current policy as the value function and then to improve the policy based on the value function. A standard algorithm of reinforcement learning is to learn the action-value function,

$$Q(s_t, a_t) = E\left[\sum_{\tau=t}^{\infty} \gamma^{(\tau-t)} r_\tau | s_t, a_t\right], \tag{1}$$

which estimates the cumulative future reward when action $a$ is taken at a state . The discount factor $0 < \gamma < 1$ is a meta-parameter that controls the time scale of prediction. The policy of the learner is then given by comparing action-values, e.g. according to Boltzman distribution

$$P(a|s_t) = \frac{\exp \beta Q(s_t, a)}{\sum_{\tilde{a} \in A} \exp \beta Q(s_t, \tilde{a})}, \tag{2}$$

where the inverse temperature $\beta > 0$ is another meta-parameter that controls randomness of action selection. From an experience of state $s_t$, action $a_t$, reward $r_t$, and next state $s_{t+1}$, the action-value function is updated by Q-learning algorithm [1] as

$$\delta_{TD}(t) = r_t + \gamma \max_{a \in A} Q(s_{t+1}, a) - Q(s_t, a_t)$$

$$Q(s_t, a_t) \leftarrow Q(s_t, a_t) + \alpha \delta_{TD}(t) \tag{3}$$

where $\alpha > 0$ is the meta-parameter that controls learning rate. Thus this simple reinforcement learning modol has three meta-paramters, $\alpha$,$\beta$ and $\gamma$ Such a reinforcement learning model does not only predict subject's actions, but also predicts internal process of the brain, which may be recorded as neural firing or brain imaging data. However, a big problem is that the predictions are depended on the setting of meta-parameters, such as learning rate $\alpha$, action randomness $\beta$ and discount factor $\gamma$.

## 3 Bayesian estimation of hidden variables of reinforcement learning agent

Let us consider a problem of estimating the time course of action-values $\{Q_t(s, a); s \in S, s \in A, 0 \leq t \leq T\}$ and meta-parameters $\alpha$, $\beta$ , and $\gamma$ of reinforcement learner by only observing the sequence of states $s_t$, actions $a_t$ and rewards $r_t$. We use a Bayesian method of estimating a dynamic hidden variable $\{\mathbf{x}_t; t \in N\}$ from sequence of observable variable $\{\mathbf{y}_t; t \in N\}$. We assume that the hidden variable follows a Markov process

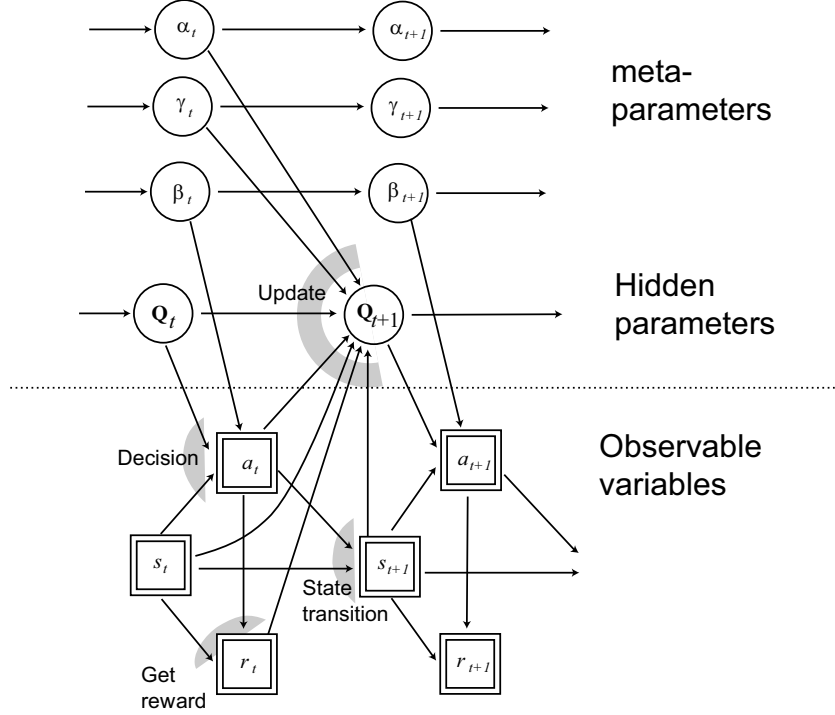

meta-parameters

Hidden parameters

Observable variables

Figure 1: A Bayesian network representation of a Q-learning agent: dynamics of observable and unobservable variable is depended on decision, reward probability, state transition, and update rule for value function. Circles: hidden variable. Double box: observable variable. Arrow: probabilistic dependency

of initial distribution $p(\mathbf{x}_0)$ and the transition probability $p(\mathbf{x}_{t+1}|\mathbf{x}_t)$. The observations $\{\mathbf{y}_t; t \in N\}$ are assumed to be conditionally independent given the process $\{\mathbf{x}_t; t \in N\}$ and has the marginal distribution $p(\mathbf{y}_t|\mathbf{x}_t)$. The problem is to estimate recursively in time the posterior distribution of hidden variable $p(\mathbf{x}_{0:t}|\mathbf{y}_{1:t})$, where $\mathbf{x}_{0:t} = \{\mathbf{x}_0, \dots, \mathbf{x}_t\}$ and $\mathbf{y}_{1:t} = \{\mathbf{y}_1, \dots, \mathbf{y}_t\}$. The marginal distribution is given by recursive procedure of the following prediction and updating,

$$Predicdion \quad : \quad p(\mathbf{x}_t|\mathbf{y}_{1:t-1}) = \int p(\mathbf{x}_t|\mathbf{x}_{t-1})p(\mathbf{x}_{t-1}|\mathbf{y}_{1:t-1})d\mathbf{x}_{t-1},$$

$$Updating \quad : \quad p(\mathbf{x}_t|\mathbf{y}_{1:t}) = \frac{p(\mathbf{y}_t|\mathbf{x}_t)p(\mathbf{x}_t|\mathbf{y}_{1:t-1})}{\int p(\mathbf{y}_t|\mathbf{x}_t)p(\mathbf{x}_t|\mathbf{y}_{1:t-1})d\mathbf{x}_t}.$$

We use a numerical method called particle filter [3] to approximate this process. In the particle filter, the distributions of sequence of hidden variables $p(\mathbf{x}_{0:t}|\mathbf{y}_{1:t})$ are represented by a set of random samples, called "particles". Figure 1 is the dynamical Bayesian network representation of a Q-learning agent. The hidden variable $\mathbf{x}_t$ consists of the action-values $Q(s, a)$ for each state-action pair, learning rate $\alpha$, inverse temperature $\beta$, and discount factor $\gamma$. The observable variable $\mathbf{y}_t$ consists of the state $s_t$, action $a_t$, and reward $r_t$.

The marginal distribution $p(\mathbf{y}_t|\mathbf{x}_t)$ of observation process is given by the softmax action selection probability (2) combined with the state transition rule and the reward condition $p(r_{t+1}|s_t, a_t)$ given by the environment. The transition probability $p(s_{t+1}|s_t, a_t)$ of the hidden variable is given by the Q-learning rule (3) and an assumption about the meta-

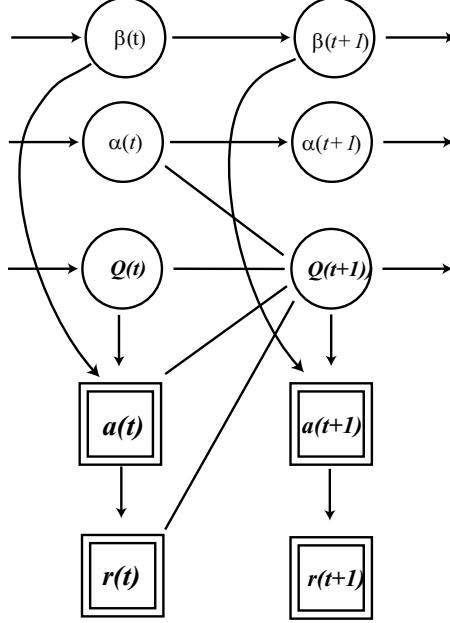

Figure 2: Simplified Bayesian network for the two-armed bandit problem.

parameter dynamics. Here we assume that meta-parameters are constant with small drifts. Because $\alpha$, $\beta$ and $\gamma$ should all be positive, we assume random-walk dynamics in logarithmic space

$$\log(x_{t+1}) = \log(x_t) + \varepsilon_x, \qquad \varepsilon_x \sim N(0, \sigma_x) \qquad (4)$$

where $\sigma_x$ is a meta-meta-parameter that defines variability of the meta-parameter $x \in \{\alpha, \beta, \gamma\}$.

## 4   Simulations

### 4.1   Two armed bandit problem with block wise reward change

In order to test the validity of the proposed method, we use a simple Q-leaning agent that learns a two armed bandit problem [1]. The task has only one state, two actions, and stochastic binary reward. The reward probability for each action is fixed in a block of 100 trials. The reward probabilities $P_{r1}$ for action $a = 1$ and $P_{r2}$ for action $a = 2$ are selected randomly from three settings; $\{P_{r1}, P_{r2}\} = \{0.1, 0.9\}, \{0.50.5\}, \{0.9, 0.1\}$ at the beginning of each block.

The Q-learning agent tries to learn reward expectation of each action and maximize reward acquired in each block. Because the task has only one state, the agent does not need to take into account next state's value, and thus, we set the discount factor as $\gamma = 0$. The Bayesian network for this example is simplified as Figure 2. Simulated actions are selected according to Boltzman distribution (2) using action-values $Q(a = 1)$ and $Q(a = 2)$, and the inverse temperature $\beta$. The action values are updated by equation (3) with the action $a_t$, reward $r_t$, and learning rate $\alpha$.

### 4.2 Result

We used 1000 particles for approximating the distribution of hidden variable $\mathbf{x} = (Q(a = 1), Q(a = 2), \log(\alpha), \log(\beta))$. We set the initial distribution of particles as Gaussian distribution with the mean $\{0, 0, 0, 0\}$ and the variance $\{1, 1, 3, 1\}$ for $\{Q(a = 1), Q(a = 2), \log(\alpha), \log(\beta)\}$, respectively. We set the meta-meta-parameters for learning rate as $\sigma_\alpha = 0.05$, and inverse temperature as $\sigma_\beta = 0.005$. The reward is $r = 5$ when delivered, and otherwise $r = 0$.

Figure 3(a) shows the simulated actions and rewards of 1000 trials by Q-learning agent with $\alpha = 0.05$ and $\beta = 1$. From this observable sequence of $\mathbf{y}_t = (s_t, a_t, r_t)$, the particle filter estimated the time course of action-values, $Q_t(a = 1)$ and $Q_t(a = 2)$, learning rate $\alpha_t$ and inverse temperature $\beta_t$. The expected values of the marginal distribution of these hidden variables (Figure 3(b)-(e) solid line) are in good agreement with the true value (Figure 3(b)-(e) dotted line) recorded in simulation. Although the initial estimates were inevitable inaccurate, the particle filter are good estimation of each variable after about 200 observations.

To test robustness of the particle filter approach, we generated behavioral sequences of Q-learners with different combinations of $\alpha = \{0.01, 0.15, 0.1, 0.5\}$ and $\beta = \{0.5, 1, 2, 4\}$, and estimated meta-parameters $\alpha$ and $\beta$. Even if we set a broad initial distribution of $\alpha$ and $\beta$, the expectation value of the estimated values are in good agreement with the true value. When the agent had the smallest learning rate $\alpha = 0.01$, the particle filter tended to underestimated $\beta$ and overestimated $\alpha$.

## 5 Application to monkey behavioral data

We applied the particle filter approach to monkey behavioral data of the two-armed bandit problem [4]. In this task, the monkey faces a lever that can be turned to either left or right. After adjusting a lever at center position and holding it for one second, the monkey turned the lever to left or right based on the reward probabilities assigned on each direction of lever turn. Probabilities [PL, PR] of reward delivery on the left and right turns, respectively were varied across three trial blocks as: [PL, PR]=[0.5, 0.5]; [0.1, 0.9]; [0.9, 0.1]. In each block, the monkeys shifted selection to the direction with higher reward probability.

We used 1000 particles and Gaussian initial distribution with the mean $(2,2,3,0)$ and the variance $(2,2,1,1)$ for $\mathbf{x} = (Q(R), Q(L), \log(\alpha), \log(\beta))$. We set the meta-meta-parameters for learning rate as $\sigma_\alpha = 0.05$, and for inverse temperature as $\sigma_\beta = 0.001$. The reward was $r = 5$ when delivered, and otherwise $r = 0$.

Figure 5(a) shows the sequence of selected actions and rewards in a day. Figure 5(b) shows the estimated action-values $Q(a = L)$ and $Q(a = R)$ for the left and right lever turns. The estimated action value $Q(L)$ for left action increased in the blocks of [PL, PR] = [0.9, 0.1], decreased in the blocks of [0.1, 0.9], and fluctuated in the blocks of [0.5, 0.5].

We tested whether the estimated action-value and meta-parameters could reproduce the action sequences. We quantified the prediction performance of action sequences by the likelihood of the action data given the estimated model,

$$L_t = \frac{1}{N - T + 1} \sum_{t=T}^{N} \log \hat{p}(a = a_t | \{a_1, r_1, \cdots, a_{t-1}, r_{t-1}\}, M, \theta_t), \tag{5}$$

where $\hat{p}(a)$ is estimated probability of action at $t$ by model $M$ and estimated parameters $\theta_t$ from the sequence of past experience $\{a_1, r_1, \cdots, a_{t-1}, r_{t-1}\}$.

Figure 6(b) shows the distribution of the likelihood computed for the action data of 74 sessions. We compared the predictability of the proposed method, Q-learning model with

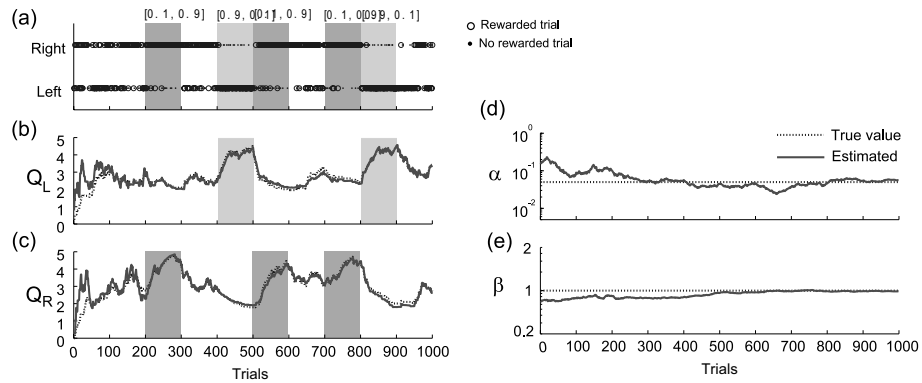

Figure 3: Estimation of hidden variables by simulated actions and rewards of Q-learning agent. (a) Sequence of simulated actions and rewards by Q-learning agent: Circles are rewarded trials. Dots are non-rewarded trials; (b)-(e) Time course of the hidden variables of the model (dotted line) and of the expectation value (solid line) of estimation by particle filter; (b)(c) Q-values for each action, (d) learning rate , and (e) action randomness . Shaded areas indicate the blocks of [0.9, 0.1] or [0.1, 0.9]. White areas indicate [0.5, 0.5].

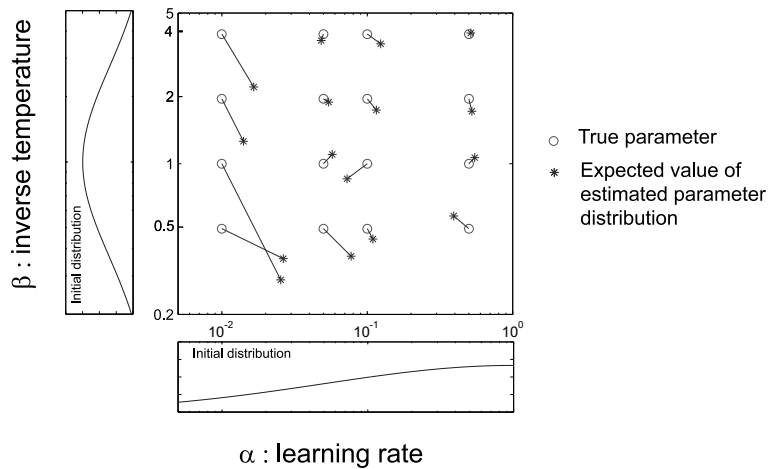

Figure 4: Expected values of estimated meta-parameter from the 1000 trials generated with different settings. The side boxes show initial distribution of particles.

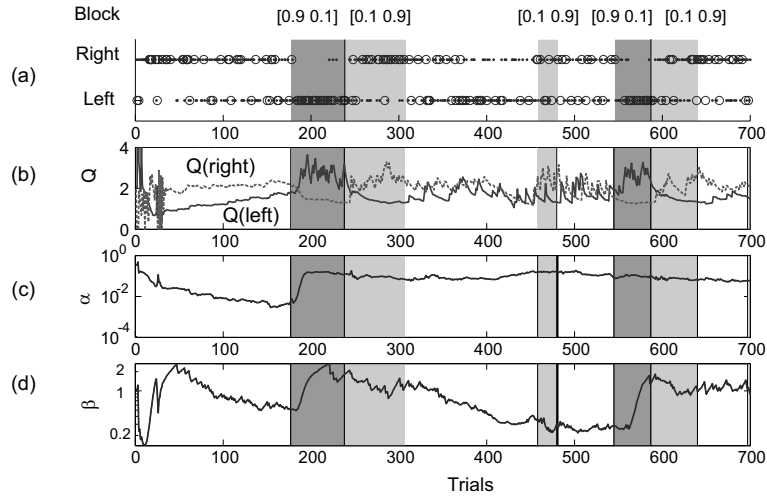

Figure 5: Expected values of estimated hidden variables by animal behavioral data: (a) action and reward sequences; Circles are rewarded trials; Dots indicate no rewarded trials. (b)-(d) Estimated value of (b) action value function , (c) learning rate, and (d) action randomness. Shaded areas indicate the blocks of [0.9, 0.1] or [0.1, 0.9]. White areas indicate [0.5, 0.5].

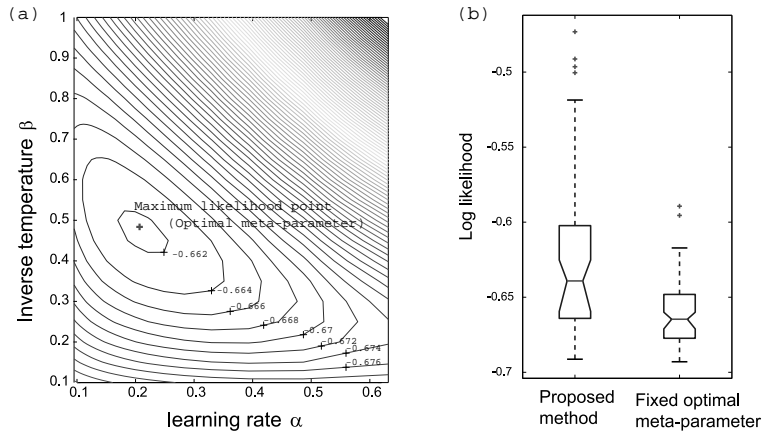

Figure 6: Comparing models: (a) An example of contour plot of log likelihood for predicted action by a fixed meta-parameter Q-learning model. Fixed meta-parameter method needs to find the optimal learning rate $\alpha$ and the inverse temperature $\beta$. (b) Distributions of log likelihood of action prediction by proposed particle filter method and by fixed meta-parameter Q-learning model with the optimal meta-parameter: The top and bottom limits of each box show the lower quartile and the upper quartile, and the center of the notch is the median. Crosses indicate outliers. Boxplot notches show the 95% confidence interval for the median. The median of log likelihood of action prediction by proposed method is significantly larger than one by the fixed meta-parameter method ( Wilcoxon signed rank test; $p < 0.0001$).

estimating meta-parameters by particle filtering, to the fixed meta-parameter Q-learning model, which used the fixed optimal learning rate $\alpha$ and inverse temperature $\beta$ in the meaning of maximizing likelihood of action prediction in a session (Figure 6(a)).

The particle filter could predict actions better than fixed meta-parameter Q-learning model with the optimal meta-parameter (Wilcoxon signed rank test; $p < 0.0001$). This result indicated that the particle filtering method successfully track the change of the meta-parameters, the learning rate $\alpha$ and the inverse temperature $\beta$, through the sessions.

# 6    Discussion

An advantage of the proposed particle filter method is that we do not have to hand-tune meta-parameter, such as learning rate. Although we still have to set the meta-meta- parameters, which defines dynamics of meta-parameters, the behavior of the estimates are less sensitive to their settings, compared to the setting of the meta-parameters. Dependency on the initial distribution of the hidden variables decreases with increasing number of data.

An extension of this study would be to model selection objectively using a hierarchical Bayesian approach. For example, the several possible reinforcement learning models, e.g. Q-learning, Sarsa algorithm or policy gradient algorithm, could be compared in term of measure of the posterior probability of models.

Recently, computational models with heuristic meta-parameters have been successfully used to generate regressors for neuroimaging data [5]. Bayesian method enables generating such regressors in a more objective, data-driven manner. We are going to apply the current method for characterizing neural recording data from the monkey.

# 7    Conclusion

We proposed a particle filter method to estimate internal parameters and meta-parameters of a reinforcement learning agent from observable variables. Our method is a powerful tool for interpreting neurophysiological and neuroimaging data in light of computational models, and to build better models in light of experimental data.

### Acknowledgments

This research was conducted as part of 'Research on Human Communication'; with funding from the Telecommunications Advancement Organization of Japan

### References

[1] Sutton RS & Barto AG (1998) *Reinforcement Learning: An Introduction*, MIT Press, Cambridge, MA.

[2] Schultz W, Dayan P, Montague PR (1997) A neural substrate of prediction and reward. *Science*. 14;275(5306):1593-1599

[3] Doucet A, de Freitas N and Gordon. N, (2001) An introduction to sequential Monte Carlo methods, In *Sequential Monte Carlo Methods in Practice*, Doucet A, de Freitas N & Gordon N eds, Springer-Verlag, pp.3-14.

[4] Ueda Y, Samejima K, Doya K, & Kimura M (2002) Reward value dependent striate neuron activity of monkey performing trial-and-error behavioral decision task, *Abst. of Soc Neurosci*, 765.13.

[5] O'Doherty, Dayan P, Friston K , Critchley H and Dolan R (2003) Temporal difference models and reward-related learning in human brain, *Neuron 28*, 329-337.
